# Linear Combinations of Optic Flow Vectors for Estimating Self-Motion –a Real-World Test of a Neural Model

**Matthias O. Franz**
MPI für biologische Kybernetik
Spemannstr. 38
D-72076 Tübingen, Germany
mof@tuebingen.mpg.de

**Javaan S. Chahl**
Center of Visual Sciences, RSBS
Australian National University
Canberra, ACT, Australia
javaan@zappa.anu.edu.au

## Abstract

The tangential neurons in the fly brain are sensitive to the typical optic flow patterns generated during self-motion. In this study, we examine whether a simplified linear model of these neurons can be used to estimate self-motion from the optic flow. We present a theory for the construction of an estimator consisting of a linear combination of optic flow vectors that incorporates prior knowledge both about the distance distribution of the environment, and about the noise and self-motion statistics of the sensor. The estimator is tested on a gantry carrying an omnidirectional vision sensor. The experiments show that the proposed approach leads to accurate and robust estimates of rotation rates, whereas translation estimates turn out to be less reliable.

## 1  Introduction

The tangential neurons in the fly brain are known to respond in a directionally selective manner to wide-field motion stimuli. A detailed mapping of their local motion sensitivities and preferred motion directions shows a striking similarity to certain self-motion-induced flow fields (an example is shown in Fig. 1). This suggests a possible involvement of these neurons in the extraction of self-motion parameters from the optic flow, which might be useful, for instance, for stabilizing the fly's head during flight manoeuvres.

A recent study [2] has shown that a simplified computational model of the tangential neurons as a weighted sum of flow measurements was able to reproduce the observed response fields. The weights were chosen according to an optimality principle which minimizes the output variance of the model caused by noise and distance variability between different scenes. The question on how the output of such processing units could be used for self-motion estimation was left open, however.

In this paper, we want to fill a part of this gap by presenting a classical linear estimation approach that extends a special case of the previous model to the complete self-motion problem. We again use linear combinations of local flow measurements but, instead of prescribing a fixed motion axis and minimizing the output variance, we require that the quadratic error in the estimated self-motion parameters be as small as possible. From this

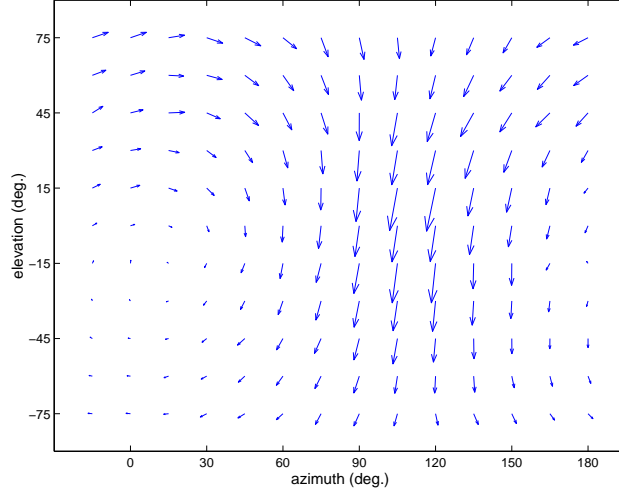

Figure 1: Mercator map of the response field of the neuron VS7. The orientation of each arrow gives the local preferred direction (LPD), and its length denotes the relative local motion sensitivity (LMS). VS7 responds maximally to rotation around an axis at an azimuth of about $30°$ and an elevation of about $-15°$ (after [1]).

optimization principle, we derive weight sets that lead to motion sensitivities similar to those observed in tangential neurons. In contrast to the previous model, this approach also yields the preferred motion directions and the motion axes to which the neural models are tuned. We subject the obtained linear estimator to a rigorous real-world test on a gantry carrying an omnidirectional vision sensor.

## 2 Modeling fly tangential neurons as optimal linear estimators for self-motion

### 2.1 Sensor and neuron model

In order to simplify the mathematical treatment, we assume that the $N$ elementary motion detectors (EMDs) of our model eye are arranged on the unit sphere. The viewing direction of a particular EMD with index $i$ is denoted by the radial unit vector $\mathbf{d}_i$. At each viewing direction, we define a local two-dimensional coordinate system on the sphere consisting of two orthogonal tangential unit vectors $\mathbf{u}_i$ and $\mathbf{v}_i$ (Fig. $2a$). We assume that we measure the local flow component along both unit vectors subject to additive noise. Formally, this means that we obtain at each viewing direction two measurements $x_i$ and $y_i$ along $\mathbf{u}_i$ and $\mathbf{v}_i$, respectively, given by

$$x_i = \mathbf{p}_i \cdot \mathbf{u}_i + n_{x,i} \quad \text{and} \quad y_i = \mathbf{p}_i \cdot \mathbf{v}_i + n_{y,i}, \tag{1}$$

where $n_{x,i}$ and $n_{y,i}$ denote additive noise components and $\mathbf{p}_i$ the local optic flow vector. When the spherical sensor translates with $\mathbf{T}$ while rotating with $\mathbf{R}$ about an axis through the origin, the self-motion-induced image flow $\mathbf{p}_i$ at $\mathbf{d}_i$ is [3]

$$\mathbf{p}_i = -\mu_i(\mathbf{T} - (\mathbf{T} \cdot \mathbf{d}_i)\mathbf{d}_i) - \mathbf{R} \times \mathbf{d}_i. \tag{2}$$

$\mu_i$ is the inverse distance between the origin and the object seen in direction $\mathbf{d}_i$, the so-called "nearness". The entire collection of flow measurements $x_i$ and $y_i$ comprises the

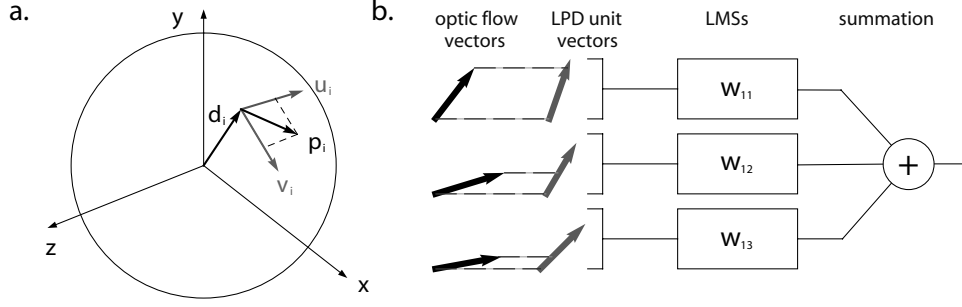

Figure 2: $a$. Sensor model: At each viewing direction $\mathbf{d}_i$, there are two measurements $x_i$ and $y_i$ of the optic flow $\mathbf{p}_i$ along two directions $\mathbf{u}_i$ and $\mathbf{v}_i$ on the unit sphere. $b$. Simplified model of a tangential neuron: The optic flow and the local noise signal are projected onto a unit vector field. The weighted projections are linearly integrated to give the estimator output.

input to the simplified neural model of a tangential neuron which consists of a weighted sum of all local measurements (Fig. 2$b$)

$$\hat{\theta} = \sum_i^N w_{x,i} x_i + \sum_i^N w_{y,i} y_i \qquad (3)$$

with local weights $w_{x,i}$ and $w_{y,i}$. In this model, the local motion sensitivity (LMS) is defined as $w_i = \|(w_{x,i}, w_{y,i})\|$, the local preferred motion direction (LPD) is parallel to the vector $\frac{1}{w_i}(w_{x,i}, w_{y,i})$. The resulting LMSs and LPDs can be compared to measurements on real tangential neurons.

As our basic hypothesis, we assume that the output of such model neurons is used to estimate the self-motion of the sensor. Since the output is a scalar, we need in the simplest case an ensemble of six neurons to encode all six rotational and translational degrees of freedom. The local weights of each neuron are chosen to yield an optimal linear estimator for the respective self-motion component.

## 2.2   Prior knowledge

An estimator for self-motion consisting of a linear combination of flow measurements necessarily has to neglect the dependence of the optic flow on the object distances. As a consequence, the estimator output will be different from scene to scene, depending on the current distance and noise characteristics. The best the estimator can do is to add up as many flow measurements as possible hoping that the individual distance deviations of the current scene from the average will cancel each other. Clearly, viewing directions with low distance variability and small noise content should receive a higher weight in this process. In this way, prior knowledge about the distance and noise statistics of the sensor and its environment can improve the reliability of the estimate.

If the current nearness at viewing direction $\mathbf{d}_i$ differs from the the average nearness $\bar{\mu}_i$ over all scenes by $\Delta\mu_i$, the measurement $x_i$ can be written as ( see Eqns. (1) and (2))

$$x_i = -(\bar{\mu}_i \mathbf{u}_i^\top, (\mathbf{u}_i \times \mathbf{d}_i)^\top) \begin{pmatrix} \mathbf{T} \\ \mathbf{R} \end{pmatrix} + n_{x,i} - \Delta\mu_i \mathbf{u}_i \mathbf{T}, \qquad (4)$$

where the last two terms vary from scene to scene, even when the sensor undergoes exactly the same self-motion.

To simplify the notation, we stack all $2N$ measurements over the entire EMD array in the vector $\mathbf{x} = (x_1, y_1, x_2, y_2, ..., x_N, y_N)^\top$. Similarly, the self-motion components along the x-, y- and z-directions of the global coordinate systems are combined in the vector $\theta = (T_x, T_y, T_z, R_x, R_y, R_z)^\top$, the scene-dependent terms of Eq. (4) in the $2N$-vector $\mathbf{n} = (n_{x,1} - \Delta\mu_1 \mathbf{u}_1 \mathbf{T}, n_{y,1} - \Delta\mu_1 \mathbf{v}_1 \mathbf{T}, ....)^\top$ and the scene-independent terms in the 6xN-matrix $F = ((-\bar{\mu}_1 \mathbf{u}_1^\top, -(\mathbf{u}_1 \times \mathbf{d}_1)^\top), (-\bar{\mu}_1 \mathbf{v}_1^\top, -(\mathbf{v}_1 \times \mathbf{d}_1)^\top), ....)^\top$. The entire ensemble of measurements over the sensor can thus be written as

$$\mathbf{x} = F\theta + \mathbf{n}. \tag{5}$$

Assuming that $\mathbf{T}$, $n_{x,i}$, $n_{y,i}$ and $\mu_i$ are uncorrelated, the covariance matrix $C$ of the scene-dependent measurement component $\mathbf{n}$ is given by

$$C_{ij} = C_{n,ij} + C_{\mu,ij} \mathbf{u}_i^\top C_T \mathbf{u}_j \tag{6}$$

with $C_n$ being the covariance of $n$, $C_\mu$ of $\mu$ and $C_T$ of $\mathbf{T}$. These three covariance matrices, together with the average nearness $\bar{\mu}_i$, constitute the prior knowledge required for deriving the optimal estimator.

## 2.3 Optimized neural model

Using the notation of Eq. (5), we write the linear estimator as

$$\hat{\theta} = W\mathbf{x}. \tag{7}$$

$W$ denotes a $2N$x6 weight matrix where each of the six rows corresponds to one model neuron (see Eq. (3)) tuned to a different component of $\theta$. The optimal weight matrix is chosen to minimize the mean square error $e$ of the estimator given by

$$e = E(\|\theta - \hat{\theta}\|^2) = tr[WCW^\top] \tag{8}$$

where $E$ denotes the expectation. We additionally impose the constraint that the estimator should be unbiased for $\mathbf{n} = 0$, i.e., $\hat{\theta} = \theta$. From Eqns. (5) and (7) we obtain the constraint equation

$$WF = \mathbf{1}_{6x6}. \tag{9}$$

The solution minimizing the associated Euler-Lagrange functional ($\Lambda$ is a 6x6-matrix of Lagrange multipliers)

$$J = tr[WCW^\top] + tr[\Lambda^\top(\mathbf{1}_{6x6} - WF)] \tag{10}$$

can be found analytically and is given by

$$W = \frac{1}{2}\Lambda F^\top C^{-1} \tag{11}$$

with $\Lambda = 2(F^\top C^{-1} F)^{-1}$. When computed for the typical inter-scene covariances of a flying animal, the resulting weight sets are able to reproduce the characteristics of the LMS and LPD distribution of the tangential neurons [2]. Having shown the good correspondence between model neurons and measurement, the question remains whether the output of such an ensemble of neurons can be used for some real-world task. This is by no means evident given the fact that - in contrast to most approaches in computer vision - the distance distribution of the current scene is completely ignored by the linear estimator.

# 3 Experiments

## 3.1 Linear estimator for an office robot

As our test scenario, we consider the situation of a mobile robot in an office environment. This scenario allows for measuring the typical motion patterns and the associated distance statistics which otherwise would be difficult to obtain for a flying agent.

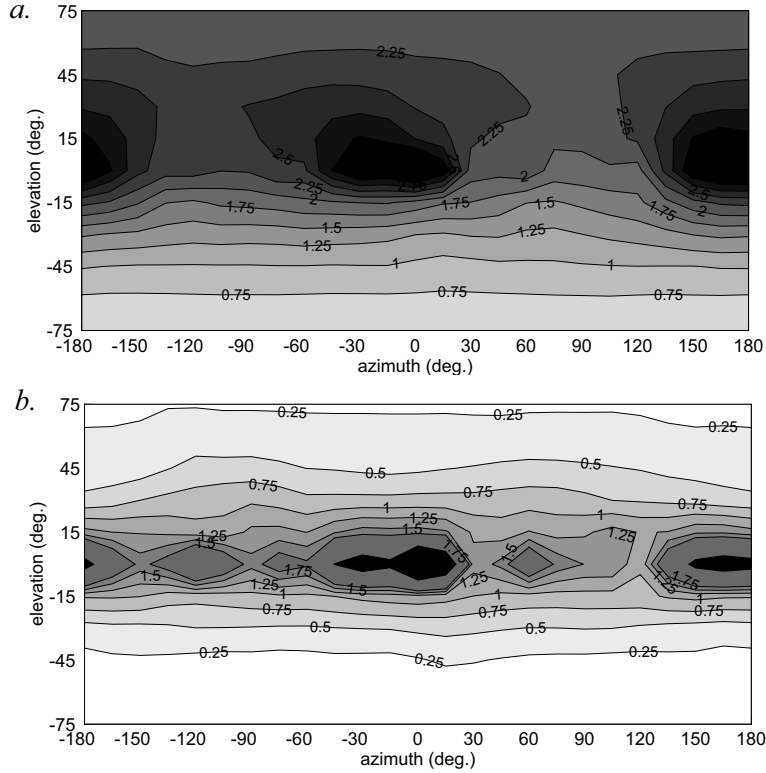

Figure 3: Distance statistics of an indoor robot (0 azimuth corresponds to forward direction): *a.* Average distances from the origin in the visual field ($N = 26$). Darker areas represent larger distances. *b.* Distance standard deviation in the visual field ($N = 26$). Darker areas represent stronger deviations.

The distance statistics were recorded using a rotating laser scanner. The 26 measurement points were chosen along typical trajectories of a mobile robot while wandering around and avoiding obstacles in an office environment. The recorded distance statistics therefore reflect properties both of the environment and of the specific movement patterns of the robot. From these measurements, the average nearness $\bar{\mu}_i$ and its covariance $C_\mu$ were computed (cf. Fig. 3, we used distance instead of nearness for easier interpretation).

The distance statistics show a pronounced anisotropy which can be attributed to three main causes: (1) Since the robot tries to turn away from the obstacles, the distance in front and behind the robot tends to be larger than on its sides (Fig. 3*a*). (2) The camera on the robot usually moves at a fixed height above ground on a flat surface. As a consequence, distance variation is particularly small at very low elevations (Fig. 3*b*). (3) The office environment also contains corridors. When the robot follows the corridor while avoiding obstacles, distance variations in the frontal region of the visual field are very large (Fig. 3*b*).

The estimation of the translation covariance $C_T$ is straightforward since our robot can only translate in forward direction, i.e. along the $z$-axis. $C_T$ is therefore 0 everywhere except the lower right diagonal entry which is the square of the average forward speed of the robot (here: 0.3 m/s). The EMD noise was assumed to be zero-mean, uncorrelated and uniform over the image, which results in a diagonal $C_n$ with identical entries. The noise standard

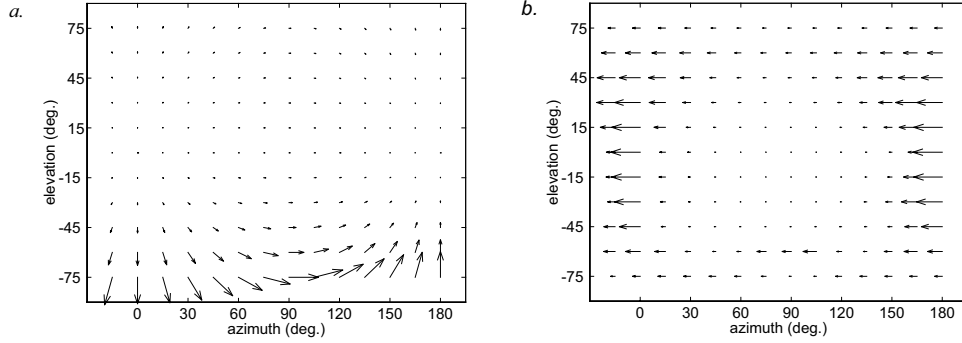

Figure 4: Model neurons computed as part of the linear estimator. Notation is identical to Fig. 1. The depicted region of the visual field extends from $-15°$ to $180°$ azimuth and from $-75°$ to $75°$ elevation. The model neurons are tuned to $a$. forward translation, and $b$. to rotations about the vertical axis.

deviation of 0.34 deg./s was determined by presenting a series of natural images moving at 1.1 deg./s to the flow algorithm used in the implementation of the estimator (see Sect. 3.2). $\bar{\mu}$, $C_\mu$, $C_T$ and $C_n$ constitute the prior knowledge necessary for computing the estimator (Eqns. (6) and (11)).

Examples of the optimal weight sets for the model neurons (corresponding to a row of $W$) are shown in Fig. 4. The resulting model neurons show very similar characteristics to those observed in real tangential neurons, however, with specific adaptations to the indoor robot scenario. All model neurons have in common that image regions near the rotation or translation axis receive less weight. In these regions, the self-motion components to be estimated generate only small flow vectors which are easily corrupted by noise. Equation (11) predicts that the estimator will preferably sample in image regions with smaller distance variations. In our measurements, this is mainly the case at the ground around the robot (Fig. 3). The rotation-selective model neurons weight image regions with larger distances more highly, since distance variations at large distances have a smaller effect. In our example, distances are largest in front and behind the robot so that the rotation-selective neurons assign the highest weights to these regions (Fig. 3$b$).

## 3.2 Gantry experiments

The self-motion estimates from the model neuron ensemble were tested on a gantry with three translational and one rotational (yaw) degree of freedom. Since the gantry had a position accuracy below 1mm, the programmed position values were taken as ground truth for evaluating the estimator's accuracy.

As vision sensor, we used a camera mounted above a mirror with a circularly symmetric hyperbolic profile. This setup allowed for a $360°$ horizontal field of view extending from $90°$ below to $45°$ above the horizon. Such a large field of view considerably improves the estimator's performance since the individual distance deviations in the scene are more likely to be averaged out. More details about the omnidirectional camera can be found in [4]. In each experiment, the camera was moved to 10 different start positions in the lab with largely varying distance distributions. After recording an image of the scene at the start position, the gantry translated and rotated at various prescribed speeds and directions and took a second image. After the recorded image pairs (10 for each type of movement) were unwarped, we computed the optic flow input for the model neurons using a standard gradient-based scheme [5].

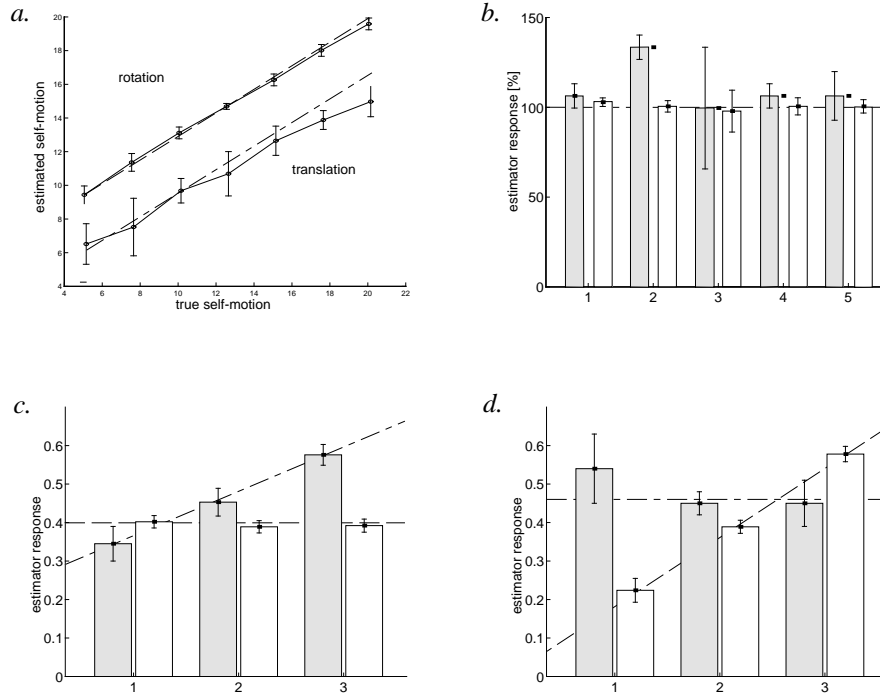

Figure 5: Gantry experiments: Results are given in arbitrary units, true rotation values are denoted by a dashed line, translation by a dash-dot line. Grey bars denote translation estimates, white bars rotation estimates *a*. Estimated vs. real self-motion; *b*. Estimates of the same self-motion at different locations; *c*. Estimates for constant rotation and varying translation; *d*. Estimates for constant translation and varying rotation.

The average error of the rotation rate estimates over all trials (N=450) was $0.7°/s$ (5.7% rel. error, Fig. 5$a$), the error in the estimated translation speeds (N=420) was 8.5 mm/s (7.5% rel. error). The estimated rotation axis had an average error of magnitude $1.7°$, the estimated translation direction $4.5°$. The larger error of the translation estimates is mainly caused by the direct dependence of the translational flow on distance (see Eq. (2)) whereas the rotation estimates are only indirectly affected by distance errors via the current translational flow component which is largely filtered out by the LPD arrangement. The larger sensitivity of the translation estimates can be seen by moving the sensor at the same translation and rotation speeds in various locations. The rotation estimates remain consistent over all locations whereas the translation estimates show a higher variance and also a location-dependent bias, e.g., very close to laboratory walls (Fig. 5$b$). A second problem for translation estimation comes from the different properties of rotational and translational flow fields: Due to its distance dependence, the translational flow field shows a much wider range of values than a rotational flow field. The smaller translational flow vectors are often swamped by simultaneous rotation or noise, and the larger ones tend to be in the upper saturation range of the used optic flow algorithm. This can be demonstrated by simultaneously translating and rotating the semsor. Again, rotation estimates remain consistent while translation estimates are strongly affected by rotation (Fig. 5$c$ and $d$).

# 4 Conclusion

Our experiments show that it is indeed possible to obtain useful self-motion estimates from an ensemble of linear model neurons. Although a linear approach necessarily has to ignore the distances of the currently perceived scene, an appropriate choice of local weights and a large field of view are capable of reducing the influence of noise and the particular scene distances on the estimates. In particular, rotation estimates were highly accurate - in a range comparable to gyroscopic estimates - and consistent across different scenes and different simultaneous translations. Translation estimates, however, turned out to be less accurate and less robust against changing scenes and simultaneous rotation.

The components of the estimator are simplified model neurons which have been shown to reproduce the essential receptive field properties of the fly's tangential neurons [2]. Our study suggests that the output of such neurons could be directly used for self-motion estimation by simply combining them linearly at a later integration stage. As our experiments have shown, the achievable accuracy would probably be more than enough for head stabilization under closed loop conditions.

Finally, we have to point out a basic limitation of the proposed theory: It assumes linear EMDs as input to the neurons (see Eq. (1)). The output of fly EMDs, however, is only linear for very small image motions. It quickly saturates at a plateau value at higher image velocities. In this range, the tangential neuron can only indicate the presence and the sign of a particular self-motion component, not the current rotation or translation velocity. A linear combination of output signals, as in our model, is no more feasible but would require some form of population coding. In addition, a detailed comparison between the linear model and real neurons shows characteristic differences indicating that tangential neurons usually operate in the plateau range rather than in the linear range of the EMDs [2]. As a consequence, our study can only give a hint on what might happen at small image velocities. The case of higher image velocities has to await further research.

### Acknowledgments

The gantry experiments were done at the Center of Visual Sciences in Canberra. The authors wish to thank J. Hill, M. Hofmann and M. V. Srinivasan for their help. Financial support was provided by the Human Frontier Science Program and the Max-Planck-Gesellschaft.

### References

[1] Krapp, H.G., Hengstenberg, B., & Hengstenberg, R. (1998). Dendritic structure and receptive field organization of optic low processing interneurons in the fly. *J. of Neurophysiology*, **79**, 1902 - 1917.

[2] Franz, M. O. & Krapp, H C. (2000). Wide-field, motion-sensitive neurons and matched filters for optic flow fields. *Biol. Cybern.*, **83**, 185 - 197.

[3] Koenderink, J. J., & van Doorn, A. J. (1987). Facts on optic flow. *Biol. Cybern.*, **56**, 247 - 254.

[4] Chahl, J. S, & Srinivasan, M. V. (1997). Reflective surfaces for panoramic imaging. *Applied Optics*, **36**(31), 8275 - 8285.

[5] Srinivasan, M. V. (1994). An image-interpolation technique for the computation of optic flow and egomotion. *Biol. Cybern.*, **71**, 401 - 415.
